# Tensor Subspace Analysis

**Xiaofei He**[1]     **Deng Cai**[2]     **Partha Niyogi**[1]
[1] Department of Computer Science, University of Chicago
{xiaofei, niyogi}@cs.uchicago.edu
[2] Department of Computer Science, University of Illinois at Urbana-Champaign
dengcai2@uiuc.edu

## Abstract

Previous work has demonstrated that the image variations of many objects (human faces in particular) under variable lighting can be effectively modeled by low dimensional linear spaces. The typical linear subspace learning algorithms include Principal Component Analysis (PCA), Linear Discriminant Analysis (LDA), and Locality Preserving Projection (LPP). All of these methods consider an $n_1 \times n_2$ image as a high dimensional vector in $\mathbb{R}^{n_1 \times n_2}$, while an image represented in the plane is intrinsically a matrix. In this paper, we propose a new algorithm called **Tensor Subspace Analysis** (TSA). TSA considers an image as the second order tensor in $\mathcal{R}^{n_1} \otimes \mathcal{R}^{n_2}$, where $\mathcal{R}^{n_1}$ and $\mathcal{R}^{n_2}$ are two vector spaces. The relationship between the column vectors of the image matrix and that between the row vectors can be naturally characterized by TSA. TSA detects the intrinsic local geometrical structure of the tensor space by learning a lower dimensional tensor subspace. We compare our proposed approach with PCA, LDA and LPP methods on two standard databases. Experimental results demonstrate that TSA achieves better recognition rate, while being much more efficient.

## 1   Introduction

There is currently a great deal of interest in appearance-based approaches to face recognition [1], [5], [8]. When using appearance-based approaches, we usually represent an image of size $n_1 \times n_2$ pixels by a vector in $\mathbb{R}^{n_1 \times n_2}$. Throughout this paper, we denote by *face space* the set of all the face images. The face space is generally a low dimensional manifold embedded in the ambient space [6], [7], [10]. The typical linear algorithms for learning such a face manifold for recognition include Principal Component Analysis (PCA), Linear Discriminant Analysis (LDA) and Locality Preserving Projection (LPP) [4].

Most of previous works on statistical image analysis represent an image by a *vector* in high-dimensional space. However, an image is intrinsically a *matrix*, or the second order tensor. The relationship between the rows vectors of the matrix and that between the column vectors might be important for finding a projection, especially when the number of training samples is small. Recently, multilinear algebra, the algebra of higher-order tensors, was applied for analyzing the multifactor structure of image ensembles [9], [11], [12]. Vasilescu and Terzopoulos have proposed a novel face representation algorithm called Tensorface [9]. Tensorface represents the set of face images by a higher-order tensor and

extends Singular Value Decomposition (SVD) to higher-order tensor data. In this way, the multiple factors related to expression, illumination and pose can be separated from different dimensions of the tensor.

In this paper, we propose a new algorithm for image (human faces in particular) representation based on the considerations of multilinear algebra and differential geometry. We call it *Tensor Subspace Analysis* (TSA). For an image of size $n_1 \times n_2$, it is represented as the second order tensor (or, matrix) in the tensor space $\mathcal{R}^{n_1} \otimes \mathcal{R}^{n_2}$. On the other hand, the face space is generally a submanifold embedded in $\mathcal{R}^{n_1} \otimes \mathcal{R}^{n_2}$. Given some images sampled from the face manifold, we can build an adjacency graph to model the local geometrical structure of the manifold. TSA finds a projection that respects this graph structure. The obtained tensor subspace provides an optimal linear approximation to the face manifold in the sense of local isometry. Vasilescu shows how to extend SVD(PCA) to higher order tensor data. We extend Laplacian based idea to tensor data.

It is worthwhile to highlight several aspects of the proposed approach here:

1. While traditional linear dimensionality reduction algorithms like PCA, LDA and LPP find a map from $\mathbb{R}^n$ to $\mathbb{R}^l$ ($l < n$), TSA finds a map from $\mathcal{R}^{n_1} \otimes \mathcal{R}^{n_2}$ to $\mathcal{R}^{l_1} \otimes \mathcal{R}^{l_2}$ ($l_1 < n_1, l_2 < n_2$). This leads to structured dimensionality reduction.

2. TSA can be performed in either supervised, unsupervised, or semi-supervised manner. When label information is available, it can be easily incorporated into the graph structure. Also, by preserving neighborhood structure, TSA is less sensitive to noise and outliers.

3. The computation of TSA is very simple. It can be obtained by solving two eigenvector problems. The matrices in the eigen-problems are of size $n_1 \times n_1$ or $n_2 \times n_2$, which are much smaller than the matrices of size $n \times n$ ($n = n_1 \times n_2$) in PCA, LDA and LPP. Therefore, TSA is much more computationally efficient in time and storage. There are few parameters that are independently estimated, so performance in small data sets is very good.

4. TSA explicitly takes into account the manifold structure of the image space. The local geometrical structure is modeled by an adjacency graph.

5. This paper is primarily focused on the second order tensors (or, matrices). However, the algorithm and analysis presented here can also be applied to higher order tensors.

## 2 Tensor Subspace Analysis

In this section, we introduce a new algorithm called *Tensor Subspace Analysis* for learning a tensor subspace which respects the geometrical and discriminative structures of the original data space.

### 2.1 Laplacian based Dimensionality Reduction

Problems of dimensionality reduction has been considered. One general approach is based on graph Laplacian [2]. The objective function of Laplacian eigenmap is as follows:

$$\min_f \sum_{ij} \left( f(\mathbf{x}_i) - f(\mathbf{x}_j) \right)^2 S_{ij}$$

where $S$ is a similarity matrix. These optimal functions are nonlinear but may be expensive to compute.

A class of algorithms may be optimized by restricting problem to more tractable families of functions. One natural approach restricts to linear function giving rise to LPP [4]. In this

paper we will consider a more structured subset of linear functions that arise out of tensor analysis. This provided greater computational benefits.

## 2.2   The Linear Dimensionality Reduction Problem in Tensor Space

The generic problem of linear dimensionality reduction in the second order tensor space is the following. Given a set of data points $X_1, \cdots, X_m$ in $\mathcal{R}^{n_1} \otimes \mathcal{R}^{n_2}$, find two transformation matrices $U$ of size $n_1 \times l_1$ and $V$ of size $n_2 \times l_2$ that maps these $m$ points to a set of points $Y_1, \cdots, Y_m \in \mathcal{R}^{l_1} \otimes \mathcal{R}^{l_2} (l_1 < n_1, l_2 < n_2)$, such that $Y_i$ "represents" $X_i$, where $Y_i = U^T X_i V$. Our method is of particular applicability in the special case where $X_1, \cdots, X_m \in \mathcal{M}$ and $\mathcal{M}$ is a nonlinear submanifold embedded in $\mathcal{R}^{n_1} \otimes \mathcal{R}^{n_2}$.

## 2.3   Optimal Linear Embeddings

As we described previously, the face space is probably a nonlinear submanifold embedded in the tensor space. One hopes then to estimate geometrical and topological properties of the submanifold from random points ("scattered data") lying on this unknown submanifold. In this section, we consider the particular question of finding a linear subspace approximation to the submanifold in the sense of local isometry. Our method is fundamentally based on LPP [4].

Given $m$ data points $\mathcal{X} = \{X_1, \cdots, X_m\}$ sampled from the face submanifold $\mathcal{M} \in \mathcal{R}^{n_1} \otimes \mathcal{R}^{n_1}$, one can build a nearest neighbor graph $\mathcal{G}$ to model the local geometrical structure of $\mathcal{M}$. Let $S$ be the weight matrix of $\mathcal{G}$. A possible definition of $S$ is as follows:

$$S_{ij} = \begin{cases} e^{-\frac{\|X_i - X_j\|^2}{t}}, & \text{if } X_i \text{ is among the } k \text{ nearest} \\ & \text{neighbors of } X_j, \text{ or } X_j \text{ is among} \\ & \text{the } k \text{ nearest neighbors of } X_i; \\ 0, & \text{otherwise.} \end{cases} \tag{1}$$

where $t$ is a suitable constant. The function $\exp(-\|X_i - X_j\|^2/t)$ is the so called heat kernel which is intimately related to the manifold structure. $\|\cdot\|$ is the Frobenius norm of matrix, i.e. $\|A\| = \sqrt{\sum_i \sum_j a_{ij}^2}$. When the label information is available, it can be easily incorporated into the graph as follows:

$$S_{ij} = \begin{cases} e^{-\frac{\|X_i - X_j\|^2}{t}}, & \text{if } X_i \text{ and } X_j \text{ share the same label;} \\ 0, & \text{otherwise.} \end{cases} \tag{2}$$

Let $U$ and $V$ be the transformation matrices. A reasonable transformation respecting the graph structure can be obtained by solving the following objective functions:

$$\min_{U,V} \sum_{ij} \|U^T X_i V - U^T X_j V\|^2 S_{ij} \tag{3}$$

The objective function incurs a heavy penalty if neighboring points $X_i$ and $X_j$ are mapped far apart. Therefore, minimizing it is an attempt to ensure that if $X_i$ and $X_j$ are "close" then $U^T X_i V$ and $U^T X_j V$ are "close" as well. Let $Y_i = U^T X_i V$. Let $D$ be a diagonal matrix, $D_{ii} = \sum_j S_{ij}$. Since $\|A\|^2 = tr(AA^T)$, we see that:

$$\frac{1}{2} \sum_{ij} \|U^T X_i V - U^T X_j V\|^2 S_{ij} = \frac{1}{2} \sum_{ij} tr\left((Y_i - Y_j)(Y_i - Y_j)^T\right) S_{ij}$$

$$= \frac{1}{2} \sum_{ij} tr\left(Y_i Y_i^T + Y_j Y_j^T - Y_i Y_j^T - Y_j Y_i^T\right) S_{ij}$$

$$= tr\left(\sum_i D_{ii} Y_i Y_i^T - \sum_{ij} S_{ij} Y_i Y_j^T\right)$$

$$= tr\left(\sum_i D_{ii} U^T X_i V V^T X_i^T U - \sum_{ij} S_{ij} U^T X_i V V^T X_j^T U\right)$$

$$= tr\left(U^T \left(\sum_i D_{ii} X_i V V^T X_i^T - \sum_{ij} S_{ij} X_i V V^T X_j^T\right) U\right)$$

$$\doteq tr\left(U^T \left(D_V - S_V\right) U\right)$$

where $D_V = \sum_i D_{ii} X_i V V^T X_i^T$ and $S_V = \sum_{ij} S_{ij} X_i V V^T X_j^T$. Similarly, $\|A\|^2 = tr(A^T A)$, so we also have

$$\frac{1}{2} \sum_{ij} \|U^T X_i V - U^T X_j V\|^2 S_{ij}$$

$$= \frac{1}{2} \sum_{ij} tr\left((Y_i - Y_j)^T (Y_i - Y_j)\right) S_{ij}$$

$$= \frac{1}{2} \sum_{ij} tr\left(Y_i^T Y_i + Y_j^T Y_j - Y_i^T Y_j - Y_j^T Y_i\right) S_{ij}$$

$$= tr\left(\sum_i D_{ii} Y_i^T Y_i - \sum_{ij} S_{ij} Y_i^T Y_j\right)$$

$$= tr\left(V^T \left(\sum_i D_{ii} X_i^T U U^T X_i - \sum_{ij} X_i^T U U^T X_j\right) V\right)$$

$$\doteq tr\left(V^T \left(D_U - S_U\right) V\right)$$

where $D_U = \sum_i D_{ii} X_i^T U U^T X_i$ and $S_U = \sum_{ij} S_{ij} X_i^T U U^T X_j$. Therefore, we should simultaneously minimize $tr\left(U^T \left(D_V - S_V\right) U\right)$ and $tr\left(V^T \left(D_U - S_U\right) V\right)$.

In addition to preserving the graph structure, we also aim at maximizing the global variance on the manifold. Recall that the variance of a random variable $x$ can be written as follows:

$$var(x) = \int_{\mathcal{M}} (x - \mu)^2 dP(x), \quad \mu = \int_{\mathcal{M}} x dP(x)$$

where $\mathcal{M}$ is the data manifold, $\mu$ is the expected value of $x$ and $dP$ is the probability measure on the manifold. By spectral graph theory [3], $dP$ can be discretely estimated by the diagonal matrix $D(D_{ii} = \sum_j S_{ij})$ on the sample points. Let $Y = U^T X V$ denote the random variable in the tensor subspace and suppose the data points have a zero mean. Thus, the *weighted* variance can be estimated as follows:

$$var(Y) = \sum_i \|Y_i\|^2 D_{ii} = \sum_i tr(Y_i^T Y_i) D_{ii} = \sum_i tr(V^T X_i^T U U^T X_i V) D_{ii}$$

$$= tr\left(V^T \left(\sum_i D_{ii} X_i^T U U^T X_i\right) V\right) = tr\left(V^T D_U V\right)$$

Similarly, $\|Y_i\|^2 = tr(Y_i Y_i^T)$, so we also have:

$$var(Y) = \sum_i tr(Y_i Y_i^T) D_{ii} = tr\left(U^T \left(\sum_i D_{ii} X_i V V^T X_i^T\right) U\right) = tr\left(U^T D_V U\right)$$

Finally, we get the following optimization problems:

$$\min_{U,V} \frac{tr\left(U^T \left(D_V - S_V\right) U\right)}{tr\left(U^T D_V U\right)} \tag{4}$$

$$\min_{U,V} \frac{tr\left(V^T\left(D_U - S_U\right)V\right)}{tr\left(V^T D_U V\right)} \quad\quad (5)$$

The above two minimization problems (4) and (5) depends on each other, and hence can not be solved independently. In the following subsection, we describe a simple computational method to solve these two optimization problems.

## 2.4 Computation

In this subsection, we discuss how to solve the optimization problems (4) and (5). It is easy to see that the optimal $U$ should be the generalized eigenvectors of $(D_V - S_V, D_V)$ and the optimal $V$ should be the generalized eigenvectors of $(D_U - S_U, D_U)$. However, it is difficult to compute the optimal $U$ and $V$ simultaneously since the matrices $D_V, S_V, D_U, S_U$ are not fixed. In this paper, we compute $U$ and $V$ iteratively as follows. We first fix $U$, then $V$ can be computed by solving the following generalized eigenvector problem:

$$(D_U - S_U)\mathbf{v} = \lambda D_U \mathbf{v} \quad\quad (6)$$

Once $V$ is obtained, $U$ can be updated by solving the following generalized eigenvector problem:

$$(D_V - S_V)\mathbf{u} = \lambda D_V \mathbf{u} \qu\quad (7)$$

Thus, the optimal $U$ and $V$ can be obtained by iteratively computing the generalized eigenvectors of (6) and (7). In our experiments, $U$ is initially set to the identity matrix. It is easy to show that the matrices $D_U, D_V, D_U - S_U$, and $D_V - S_V$ are all symmetric and positive semi-definite.

## 3 Experimental Results

In this section, several experiments are carried out to show the efficiency and effectiveness of our proposed algorithm for face recognition. We compare our algorithm with the Eigenface (PCA) [8], Fisherface (LDA) [1], and Laplacianface (LPP) [5] methods, three of the most popular linear methods for face recognition.

Two face databases were used. The first one is the PIE (Pose, Illumination, and Experience) database from CMU, and the second one is the ORL database. In all the experiments, preprocessing to locate the faces was applied. Original images were normalized (in scale and orientation) such that the two eyes were aligned at the same position. Then, the facial areas were cropped into the final images for matching. The size of each cropped image in all the experiments is $32 \times 32$ pixels, with 256 gray levels per pixel. No further preprocessing is done. For the Eigenface, Fisherface, and Laplacianface methods, the image is represented as a 1024-dimensional vector, while in our algorithm the image is represented as a $(32 \times 32)$-dimensional matrix, or the second order tensor. The nearest neighbor classifier is used for classification for its simplicity.

In short, the recognition process has three steps. First, we calculate the face subspace from the training set of face images; then the new face image to be identified is projected into $d$-dimensional subspace (PCA, LDA, and LPP) or $(d \times d)$-dimensional tensor subspace (TSA); finally, the new face image is identified by nearest neighbor classifier. In our TSA algorithm, the number of iterations is taken to be 3.

### 3.1 Experiments on PIE Database

The CMU PIE face database contains 68 subjects with 41,368 face images as a whole. The face images were captured by 13 synchronized cameras and 21 flashes, under varying pose, illumination and expression. We choose the five near frontal poses (C05, C07, C09, C27,

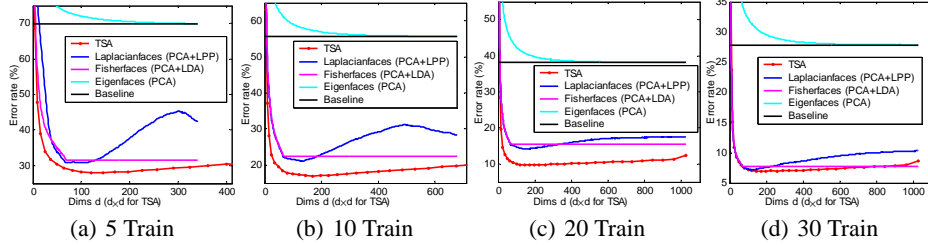

| (a) 5 Train | (b) 10 Train | (c) 20 Train | (d) 30 Train |

Figure 1: Error rate vs. dimensionality reduction on PIE database

Table 1: Performance comparison on PIE database

| Method | 5 Train | | | 10 Train | | |
|---|---|---|---|---|---|---|
| | error | dim | time(s) | error | dim | time(s) |
| Baseline | 69.9% | 1024 | - | 55.7% | 1024 | - |
| Eigenfaces | 69.9% | 338 | 0.907 | 55.7% | 654 | 5.297 |
| Fisherfaces | 31.5% | 67 | 1.843 | 22.4% | 67 | 9.609 |
| Laplacianfaces | 30.8% | 67 | 2.375 | 21.1% | 134 | 11.516 |
| TSA | **27.9%** | $\mathbf{11^2}$ | **0.594** | **16.9%** | $\mathbf{13^2}$ | **2.063** |
| | 20 Train | | | 30 Train | | |
| Method | error | dim | time(s) | error | dim | time(s) |
| Baseline | 38.2% | 1024 | - | 27.9% | 1024 | - |
| Eigenfaces | 38.1% | 889 | 14.328 | 27.9% | 990 | 15.453 |
| Fisherfaces | 15.4% | 67 | 35.828 | 7.77% | 67 | 38.406 |
| Laplacianfaces | 14.1% | 146 | 39.172 | 7.13% | 131 | 47.610 |
| TSA | **9.64%** | $\mathbf{13^2}$ | **7.125** | **6.88%** | $\mathbf{12^2}$ | **15.688** |

C29) and use all the images under different illuminations and expressions, thus we get 170 images for each individual. For each individual, $l(= 5, 10, 20, 30)$ images are randomly selected for training and the rest are used for testing.

The training set is utilized to learn the subspace representation of the face manifold by using Eigenface, Fisherface, Laplacianface and our algorithm. The testing images are projected into the face subspace in which recognition is then performed. For each given $l$, we average the results over 20 random splits. It would be important to note that the Laplacianface algorithm and our algorithm share the same graph structure as defined in Eqn. (2).

Figure 1 shows the plots of error rate versus dimensionality reduction for the Eigenface, Fisherface, Laplacianface, TSA and baseline methods. For the baseline method, the recognition is simply performed in the original 1024-dimensional image space without any dimensionality reduction. Note that, the upper bound of the dimensionality of Fisherface is $c - 1$ where $c$ is the number of individuals. For our TSA algorithm, we only show its performance in the $(d \times d)$-dimensional tensor subspace, say, 1, 4, 9, etc. As can be seen, the performance of the Eigenface, Fisherface, Laplacianface, and TSA algorithms varies with the number of dimensions. We show the best results obtained by them in Table 1 and the corresponding face subspaces are called optimal face subspace for each method.

It is found that our method outperforms the other four methods with different numbers of training samples (5, 10, 20, 30) per individual. The Eigenface method performs the worst. It does not obtain any improvement over the baseline method. The Fisherface and Laplacianface methods perform comparatively to each each. The dimensions of the optimal subspaces are also given in Table 1.

As we have discussed, TSA can be implemented very efficiently. We show the running time in seconds for each method in Table 1. As can be seen, TSA is much faster than the

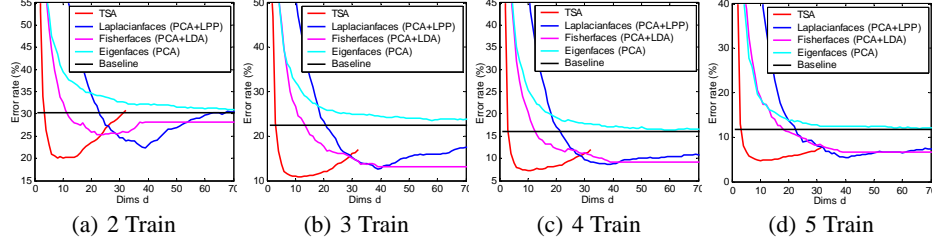

| (a) 2 Train | (b) 3 Train | (c) 4 Train | (d) 5 Train |

Figure 2: Error rate vs. dimensionality reduction on ORL database

Table 2: Performance comparison on ORL database

| Method | 2 Train | | | 3 Train | | |
|---|---|---|---|---|---|---|
| | error | dim | time | error | dim | time |
| Baseline | 30.2% | 1024 | - | 22.4% | 1024 | - |
| Eigenfaces | 30.2% | 79 | 38.13 | 22.3% | 113 | 85.16 |
| Fisherfaces | 25.2% | 23 | 60.32 | 13.1% | 39 | 119.69 |
| Laplacianfaces | 22.2% | 39 | 62.65 | 12.5% | 39 | 136.25 |
| **TSA** | **20.0%** | $\mathbf{10^2}$ | **65.00** | **10.7%** | $\mathbf{11^2}$ | **135.93** |

| | 4 Train | | | 5 Train | | |
|---|---|---|---|---|---|---|
| Method | error | dim | time | error | dim | time |
| Baseline | 16.0% | 1024 | - | 11.7% | 1024 | - |
| Eigenfaces | 15.9% | 122 | 141.72 | 11.6% | 182 | 224.69 |
| Fisherfaces | 9.17% | 39 | 212.82 | 6.55% | 39 | 355.63 |
| Laplacianfaces | 8.54% | 39 | 248.90 | 5.45% | 40 | 410.78 |
| **TSA** | **7.12%** | $\mathbf{10^2}$ | **201.40** | **4.75%** | $\mathbf{10^2}$ | **302.97** |

Eigenface, Fisherface and Laplacianface methods. All the algorithms were implemented in Matlab 6.5 and run on a Intel P4 2.566GHz PC with 1GB memory.

## 3.2 Experiments on ORL Database

The ORL (Olivetti Research Laboratory) face database is used in this test. It consists of a total of 400 face images, of a total of 40 people (10 samples per person). The images were captured at different times and have different variations including expressions (open or closed eyes, smiling or non-smiling) and facial details (glasses or no glasses). The images were taken with a tolerance for some tilting and rotation of the face up to 20 degrees. For each individual, $l(= 2, 3, 4, 5)$ images are randomly selected for training and the rest are used for testing.

The experimental design is the same as that in the last subsection. For each given $l$, we average the results over 20 random splits. Figure 3.2 shows the plots of error rate versus dimensionality reduction for the Eigenface, Fisherface, Laplacianface, TSA and baseline methods. Note that, the presentation of the performance of the TSA algorithm is different from that in the last subsection. Here, for a given $d$, we show its performance in the $(d \times d)$-dimensional tensor subspace. The reason is for better comparison, since the Eigenface and Laplacianface methods start to converge after 70 dimensions and there is no need to show their performance after that. The best result obtained in the optimal subspace and the running time (millisecond) of computing the eigenvectors for each method are shown in Table 2.

As can be seen, our TSA algorithm performed the best in all the cases. The Fisherface and Laplacianface methods performed comparatively to our method, while the Eigenface method performed poorly.

# 4 Conclusions and Future Work

Tensor based face analysis (representation and recognition) is introduced in this paper in order to detect the underlying nonlinear face manifold structure in the manner of tensor subspace learning. The manifold structure is approximated by the adjacency graph computed from the data points. The optimal tensor subspace respecting the graph structure is then obtained by solving an optimization problem. We call this *Tensor Subspace Analysis* method.

Most of traditional appearance based face recognition methods (i.e. Eigenface, Fisherface, and Laplacianface) consider an image as a vector in high dimensional space. Such representation ignores the spacial relationships between the pixels in the image. In our work, an image is naturally represented as a matrix, or the second order tensor. Tensor representation makes our algorithm much more computationally efficient than PCA, LDA, and LPP. Experimental results on PIE and ORL databases demonstrate the efficiency and effectiveness of our method.

TSA is linear. Therefore, if the face manifold is highly nonlinear, it may fail to discover the intrinsic geometrical structure. It remains unclear how to generalize our algorithm to nonlinear case. Also, in our algorithm, the adjacency graph is induced from the local geometry and class information. Different graph structures lead to different projections. It remains unclear how to define the optimal graph structure in the sense of discrimination.

## References

[1] P.N. Belhumeur, J.P. Hepanha, and D.J. Kriegman, "Eigenfaces vs. fisherfaces: recognition using class specific linear projection,"*IEEE. Trans. Pattern Analysis and Machine Intelligence*, vol. 19, no. 7, pp. 711-720, July 1997.

[2] M. Belkin and P. Niyogi, "Laplacian Eigenmaps and Spectral Techniques for Embedding and Clustering ," *Advances in Neural Information Processing Systems 14*, 2001.

[3] Fan R. K. Chung, *Spectral Graph Theory,* Regional Conference Series in Mathematics, number 92, 1997.

[4] X. He and P. Niyogi, "Locality Preserving Projections,"*Advance in Neural Information Processing Systems 16*, Vancouver, Canada, December 2003.

[5] X. He, S. Yan, Y. Hu, P. Niyogi, and H.-J. Zhang, "Face Recognition using Laplacianfaces,"*IEEE. Trans. Pattern Analysis and Machine Intelligence*, vol. 27, No. 3, 2005.

[6] S. Roweis, and L. K. Saul, "Nonlinear Dimensionality Reduction by Locally Linear Embedding," *Science*, vol 290, 22 December 2000.

[7] J. B. Tenenbaum, V. de Silva, and J. C. Langford, "A Global Geometric Framework for Nonlinear Dimensionality Reduction," *Science*, vol 290, 22 December 2000.

[8] M. Turk and A. Pentland, "Eigenfaces for recognition," *Journal of Cognitive Neuroscience*, 3(1):71-86, 1991.

[9] M. A. O. Vasilescu and D. Terzopoulos, "Multilinear Subspace Analysis for Image Ensembles," *IEEE Conference on Computer Vision and Pattern Recognition*, 2003.

[10] K. Q. Weinberger and L. K. Saul, "Unsupervised Learning of Image Manifolds by SemiDefinite Programming," *IEEE Conference on Computer Vision and Pattern Recognition*, Washington, DC, 2004.

[11] J. Yang, D. Zhang, A. Frangi, and J. Yang, "Two-dimensional PCA: a new approach to appearance-based face representation and recognition,"*IEEE. Trans. Pattern Analysis and Machine Intelligence*, vol. 26, No. 1, 2004.

[12] J. Ye, R. Janardan, Q. Li, "Two-Dimensional Linear Discriminant Analysis ," *Advances in Neural Information Processing Systems 17*, 2004.
